# Efficient Methods for Dealing with Missing Data in Supervised Learning

**Volker Tresp**[*]
Siemens AG
Central Research
Otto-Hahn-Ring 6
81730 München
Germany

**Ralph Neuneier**
Siemens AG
Central Research
Otto-Hahn-Ring 6
81730 München
Germany

**Subutai Ahmad**
Interval Research Corporation
1801-C Page Mill Rd.
Palo Alto, CA 94304

## Abstract

We present efficient algorithms for dealing with the problem of missing inputs (incomplete feature vectors) during training and recall. Our approach is based on the approximation of the input data distribution using Parzen windows. For recall, we obtain closed form solutions for arbitrary feedforward networks. For training, we show how the backpropagation step for an incomplete pattern can be approximated by a weighted averaged backpropagation step. The complexity of the solutions for training and recall is independent of the number of missing features. We verify our theoretical results using one classification and one regression problem.

## 1 Introduction

The problem of missing data (incomplete feature vectors) is of great practical and theoretical interest. In many applications it is important to know how to react if the available information is incomplete, if sensors fail or if sources of information become

---

[*] At the time of the research for this paper, a visiting researcher at the Center for Biological and Computational Learning, MIT. E-mail: Volker.Tresp@zfe.siemens.de

unavailable. As an example, when a sensor fails in a production process, it might not be necessary to stop everything if sufficient information is implicitly contained in the remaining sensor data. Furthermore, in economic forecasting, one might want to continue to use a predictor even when an input variable becomes meaningless (for example, due to political changes in a country). As we have elaborated in earlier papers, heuristics such as the substitution of the mean for an unknown feature can lead to solutions that are far from optimal (Ahmad and Tresp, 1993, Tresp, Ahmad, and Neuneier, 1994). Biological systems must deal continuously with the problem of unknown uncertain features and they are certainly extremely good at it. From a biological point of view it is therefore interesting which solutions to this problem can be derived from theory and if these solutions are in any way related to the way that biology deals with this problem (compare Brunelli and Poggio, 1991). Finally, having efficient methods for dealing with missing features allows a novel pruning strategy: if the quality of the prediction is not affected if an input is pruned, we can remove it and use our solutions for prediction with missing inputs or retrain the model without that input (Tresp, Hollatz and Ahmad, 1995).

In Ahmad and Tresp (1993) and in Tresp, Ahmad and Neuneier (1994) equations for training and recall were derived using a probabilistic setting (compare also Buntine and Weigend, 1991, Ghahramani and Jordan, 1994). For general feedforward neural networks the solution was in the form of an integral which has to be approximated using numerical integration techniques. The computational complexity of these solutions grows exponentially with the number of missing features. In these two publications, we could only obtain efficient algorithms for networks of normalized Gaussian basis functions. It is of great practical interest to find efficient ways of dealing with missing inputs for general feedforward neural networks which are more commonly used in applications. In this paper we describe an efficient approximation for the problem of missing information that is applicable to a large class of learning algorithms, including feedforward networks. The main results are Equation 2 (recall) and Equation 3 (training). One major advantage of the proposed solution is that the complexity does not increase with an increasing number of missing inputs. The solutions can easily be generalized to the problem of uncertain (noisy) inputs.

## 2   Missing Information During Recall

### 2.1   Theory

We assume that a neural network $NN(x)$ has been trained to predict $E(y|x)$, the expectation of $y \in \Re$ given $x \in \Re^D$. During recall we would like to know the network's prediction based on an incomplete input vector $x = (x^c, x^u)$ where $x^c$ denotes the known inputs and $x^u$ the unknown inputs. The optimal prediction given the known features can be written as (Ahmad and Tresp, 1993)

$$E(y|x^c) = \int E(y|x^c, x^u)P(x^u|x^c)\,dx^u \approx \frac{1}{P(x^c)} \int NN(x^c, x^u)P(x^c, x^u)\,dx^u.$$

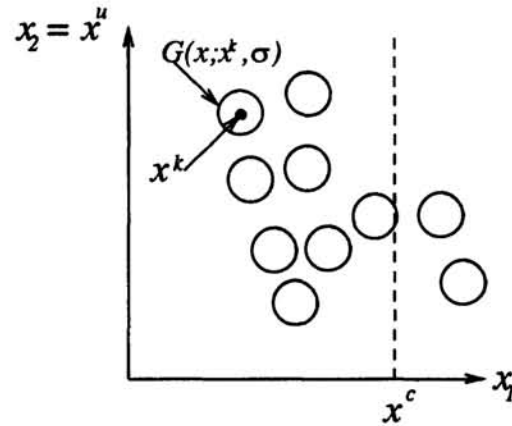

Figure 1: The circles indicate 10 Gaussians approximating the input density distribution. $x^c = x_1$ indicates the known input, $x_2 = x^u$ is unknown.

Similarly, for a network trained to estimate class probabilities, $NN_i(x) \approx P(class_i|x)$, simply substitute $P(class_i|x^c)$ for $E(y|x^c)$ and $NN_i(x^c, x^u)$ for $NN(x^c, x^u)$ in the last equation.

The integrals in the last equations can be problematic. In the worst case they have to be approximated numerically (Tresp, Ahmad and Neuneier, 1994) which is costly, since the computation is exponential in the number of missing inputs. For networks of normalized Gaussians, there exist closed form solutions to the integrals (Ahmad and Tresp, 1993). The following section shows how to efficiently approximate the integral for a large class of algorithms.

## 2.2 An Efficient Approximation

Parzen windows are commonly used to approximate densities. Given $N$ training data $\{(x^k, y^k)|k = 1, ..., N\}$, we can approximate

$$P(x) \approx \frac{1}{N} \sum_{k=1}^{N} G(x; x^k, \sigma) \tag{1}$$

where

$$G(x; x^k, \sigma) = \frac{1}{(2\pi\sigma^2)^{D/2}} \exp(-\frac{1}{2\sigma^2}||x - x^k||^2)$$

is a multidimensional properly normalized Gaussian centered at data $x^k$ with variance $\sigma^2$. It has been shown (Duda and Hart (1973)) that Parzen windows approximate densities for $N \to \infty$ arbitrarily well, if $\sigma$ is appropriately scaled.

Using Parzen windows we may write

$$E(y|x^c) \approx \frac{1}{\sum_{k=1}^{N} G(x^c; x^{c,k}, \sigma)} \sum_{k=1}^{N} [\int NN(x^c, x^u) \, G(x^c, x^u; x^k, \sigma) \, dx^u]$$

where we have used the fact that

$$P(x^c) \approx \frac{1}{N} \sum_{k=1}^{N} G(x^c; x^{c,k}, \sigma)$$

and where $G(x^c; x^{c,k}, \sigma)$ is a Gaussian projected onto the known input dimensions (by simply leaving out the unknown dimensions in the exponent and in the normalization, see Ahmad and Tresp, 1993). $x^{c,k}$ are the components of the training data corresponding to the known input (compare Figure 1).

Now, if we assume that the network prediction is approximately constant over the "width" of the Gaussians, $\sigma$, we can approximate

$$\int NN(x^c, x^u) \, G(x^c, x^u; x^k, \sigma) \, dx^u \approx NN(x^c, x^{u,k}) \, G(x^c; x^{c,k}, \sigma)$$

where $NN(x^c, x^{u,k})$ is the network prediction which we obtain if we substituted the corresponding components of the training data for the unknown inputs.

With this approximation,

$$E(y|x^c) \approx \frac{\sum_{k=1}^{N} \alpha_k \, G(x^c; x^{c,k}, \sigma)}{\sum_{k=1}^{N} G(x^c; x^{c,k}, \sigma)}, \quad \alpha_k = NN(x^c, x^{u,k}). \qquad (2)$$

Interestingly, we have obtained a network of normalized Gaussians which are centered at the known components of the data points. The "output weights" $NN(x^c, x^{u,k})$ consist of the neural network predictions where for the unknown input the corresponding components of the training data points have been substituted. Note, that we have obtained an approximation which has the same structure as the solution for normalized Gaussian basis functions (Ahmad and Tresp, 1994).

In many applications it might be easy to select a reasonable value for $\sigma$ using prior knowledge but there are also two simple ways to obtain a good estimate for $\sigma$ using leave-one-out methods. The first method consists of removing the $k - th$ pattern from the training data and calculating $\tilde{P}(x^k) \approx \frac{1}{N-1} \sum_{l=1, l \neq k}^{N} G(x^k; x^l, \sigma)$. Then select the $\sigma$ for which the log likelihood $\sum_k \log \tilde{P}(x^k)$ is maximum. The second method consists of treating an input of the $k - th$ training pattern as missing and then testing how well our algorithm (Equation 2) can predict the target. Select the $\sigma$ which gives the best performance. In this way it would even be possible to select input-dimension-specific widths $\sigma_i$ leading to "elliptical", axis-parallel Gaussians (Ahmad and Tresp, 1993).

Note that the complexity of the solution is independent of the number of missing inputs! In contrast, the complexity of the solution for feedforward networks suggested in Tresp, Ahmad and Neuneier (1994) grows exponentially with the number of missing inputs. Although similar in character to the solution for normalized RBFs, here we have no restrictions on the network architecture which allows us to choose the network most appropriate for the application.

If the amount of training data is large, one can use the following approximations:

- Select only the $K$ nearest data points. The distance is determined based on the known inputs. $K$ can probably be reasonably small ($< 10$). In the extreme case, $K = 1$ and we obtain a nearest-neighbor solution. Efficient tree-based algorithms exist for computing the $K$-nearest neighbors.

- Use Gaussian mixtures instead of Parzen windows to estimate the input data distribution. Use the centers and variances of the components in Equation 2.

- Use a clustering algorithm and use the cluster centers instead of the data points in Equation 2.

Note that the solution which substitutes the components of the training data closest to the input seems biologically plausible.

## 2.3   Experimental Results

We tested our algorithm using the same data as in Ahmad and Tresp, 1993. The task was to recognize a hand gesture based on its 2D projection. As input, the classifier is given the 2D polar coordinates of the five finger tip positions relative to the 2D center of mass of the hand (the input space is therefore 10-D). A multi-layer perceptron was trained on 4368 examples (624 poses for each gesture) and tested on a similar independent test set. The inputs were normalized to a variance of one and $\sigma$ was set to 0.1. (For a complete description of the task see (Ahmad and Tresp, 1993).) As in (Ahmad & Tresp, 1993) we defined a correct classification as one in which the correct class was either classified as the most probable or the second most probable. Figure 2 shows experimental results. On the horizontal axis, the number of randomly chosen missing inputs is shown. The continuous line shows the performance using Equation 2 where we used only the 10 nearest neighbors in the approximation. Even with 5 missing inputs we obtain a score of over 90 % which is slightly better than the solution we obtained in Ahmad and Tresp (1993) for normalized RBFs. We expect our new solution to perform very well in general since we can always choose the best network for prediction and are not restricted in the architecture. As a benchmark we also included the case where the mean of the missing input was substituted. With 5 missing inputs, the performance is less than 60 %.

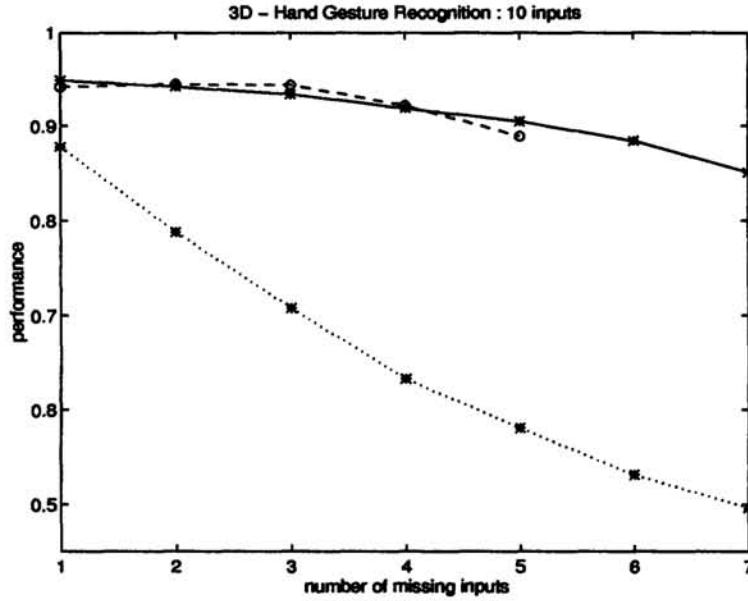

Figure 2: Experimental results using a generalization data set. The continuous line indicates the performance using our proposed method. The dotted lines indicate the performance if the mean of the missing input variable is substituted. As a comparison, we included the results obtained in Ahmad and Tresp (1993) using the closed-form solution for RBF-networks (dashed).

## 3   Training (Backpropagation)

For a complete pattern $(x^k, y^k)$, the weight update of a backpropagation step for weight $w_j$ is

$$\Delta w_j \propto (y^k - NN_w(x^k)) \frac{\partial NN_w(x^k)}{\partial w_j}.$$

Using the approximation of Equation 1, we obtain for an incomplete data point (compare Tresp, Ahmad and Neuneier, 1994)

$$\Delta w_j \propto \frac{\sum_{l \in compl} \alpha_l \; G(y^k; NN_w(x^{c,k}, x^{u,l}), \sigma_y) \; G(x^{c,k}; x^{c,l}, \sigma)}{\sum_{l \in compl} G(y^k; NN_w(x^{c,k}, x^{u,l}), \sigma_y) \; G(x^{c,k}; x^{c,l}, \sigma)}. \qquad (3)$$

Here, $l \in compl$ indicates the sum over complete patterns in the training set, and $\sigma_y$ is the standard deviation of the output noise. Note that the gradient is a network of normalized Gaussian basis functions where the "output-weight" is now

$$\alpha_l = (y^k - NN_w(x^{c,k} x^{u,l})) \; \frac{\partial NN_w(x^{c,k} x^{u,l})}{\partial w_j}$$

The derivation of the last equation can be found in the Appendix. Figure 3 shows experimental results.

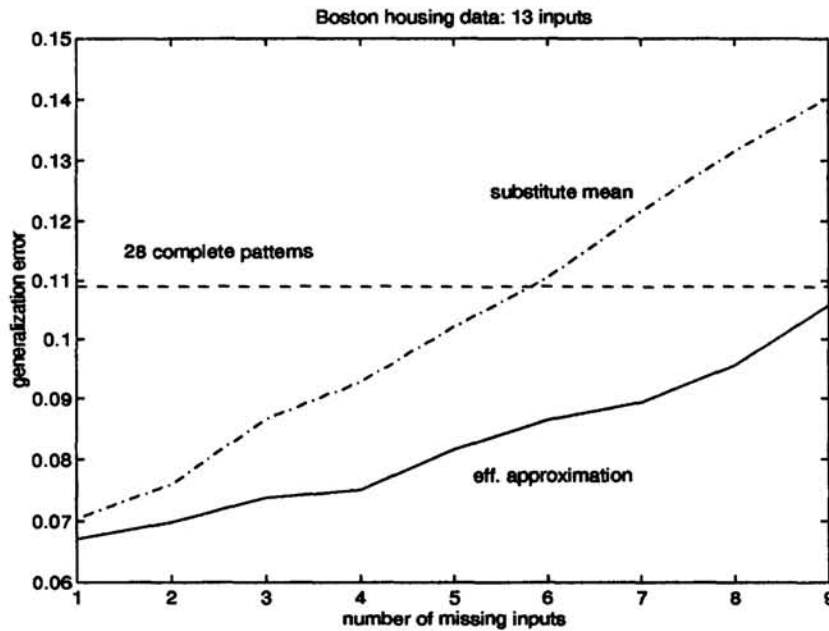

Figure 3: In the experiment, we used the Boston housing data set, which consists of 506 samples. The task is to predict the housing price from 13 variables which were thought to influence the housing price in a neighborhood. The network (multi-layer perceptron) was trained with 28 complete patterns plus an additional 225 incomplete samples. The horizontal axis indicates how many inputs were missing in these 225 samples. The vertical axis shows the generalization performance. The continuous line indicates the performance of our approach and the dash-dotted line indicates the performance, if the mean is substituted for a missing variable. The dashed line indicates the performance of a network only trained with the 28 complete patterns.

## 4  Conclusions

We have obtained efficient and robust solutions for the problem of recall and training with missing data. Experimental results verified our method. All of our results can easily be generalized to the case of noisy inputs.

### Acknowledgement

Valuable discussions with Hans-Georg Zimmermann, Tomaso Poggio, Michael Jordan and Zoubin Ghahramani are greatfully acknowledged. The first author would like to thank the Center for Biological and Computational Learning (MIT) for providing and excellent research environment during the summer of 1994.

## 5  Appendix

Assuming the standard signal-plus-Gaussian-noise model we obtain for a complete sample

$$P(x^k, y^k | \{w_i\}) = G(y^k; NN_w(x^k), \sigma_y) \, P(x^k)$$

where $\{w_i\}$ is the set of weights in the network. For an incomplete sample

$$P(x^{c,k}, y^k | \{w_i\}) = \frac{1}{P(x^{c,k}, y^k)} \int G(y^k; NN_w(x^{c,k}, x^u), \sigma_y) \, P(x^{c,k}, x^u) \, dx^u.$$

Using the same approximation as in Section 2.2,

$$P(x^{c,k}, y^k | \{w_i\}) \approx \sum_{l \in compl} G(y^k; NN_w(x^{c,k}, x^{u,l}), \sigma_y) \, G(x^{c,k}; x^{c,l}, \sigma)$$

where $l$ sums over all complete samples. As before, we substitute for the missing components the ones from the complete training data. The log-likelihood $\mathcal{L}$ (a function of the network weights $\{w_i\}$) can be calculated as ($x^k$ can be either complete or incomplete) $\mathcal{L} = \sum_{k=1}^{N} \log P(x^k, y^k | \{w_i\})$. The maximum likelihood solution consists of finding weights $\{w_i\}$ which maximize the log-likelihood. Using the approximation of Equation 1, we obtain for an incomplete sample as gradient Equation 3 (compare Tresp, Ahmad and Neuneier, 1994).

## References

Ahmad, S. and Tresp, V. (1993). Some Solutions to the Missing Feature Problem in Vision. In S. J. Hanson, J. D. Cowan and C. L. Giles, (Eds.), *Advances in Neural Information Processing Systems 5*, San Mateo, CA: Morgan Kaufmann.

Brunelli, R. and Poggio, T. (1991). *HyperBF Networks for Real Object Recognition.* IJCAI.

Buntine, W. L. and Weigend, A. S. (1991). Bayesian Back-Propagation. *Complex systems,* Vol. 5, pp. 605-643.

Duda, R. O. and Hart, P. E. (1973). *Pattern Classification and Scene Analysis.* John Wiley and Sons, New York.

Ghahramani, Z. and Jordan, M. I. (1994). Supervised Learning from Incomplete Data via an EM approach. In: Cowan, J. D., Tesauro, G., and Alspector, J., eds., *Advances in Neural Information Processing Systems 6,* San Mateo, CA, Morgan Kaufman.

Tresp, V., Ahmad, S. and Neuneier, R. (1994). Training Neural Networks with Deficient Data. In: Cowan, J. D., Tesauro, G., and Alspector, J., eds., *Advances in Neural Information Processing Systems 6,* San Mateo, CA, Morgan Kaufman.

Tresp, V., Hollatz, J. and Ahmad, S. (1995). Representing Probabilistic Rules with Networks of Gaussian Basis Functions. Accepted for publication in *Machine Learning.*
